# Scalable Discriminative Learning
# for Natural Language Parsing and Translation

**Joseph Turian, Benjamin Wellington, and I. Dan Melamed**
{lastname}@cs.nyu.edu
Computer Science Department
New York University
New York, New York 10003

## Abstract

Parsing and translating natural languages can be viewed as problems of predicting tree structures. For machine learning approaches to these predictions, the diversity and high dimensionality of the structures involved mandate very large training sets. This paper presents a purely discriminative learning method that scales up well to problems of this size. Its accuracy was at least as good as other comparable methods on a standard parsing task. To our knowledge, it is the first purely discriminative learning algorithm for translation with tree-structured models. Unlike other popular methods, this method does not require a great deal of feature engineering *a priori*, because it performs feature selection over a compound feature space as it learns. Experiments demonstrate the method's versatility, accuracy, and efficiency. Relevant software is freely available at http://nlp.cs.nyu.edu/parser and http://nlp.cs.nyu.edu/GenPar.

## 1  Introduction

Discriminative machine learning methods have led to better solutions for many problems in natural language processing (NLP), such as various kinds of sequence labeling. However, only limited advances have been made on NLP problems involving tree-structured prediction. State of the art methods for both parsing and translation use discriminative methods, but they are still limited by their reliance on generative models that can be estimated relatively cheaply. For example, some parsers and translators use a generative model to generate a list of candidates, and then rerank them using a discriminative reranker (e.g., Henderson, 2004; Charniak & Johnson, 2005; Cowan et al., 2006). Others use a generative model as a feature in a discriminative framework, because otherwise training is impractically slow (Collins & Roark, 2004; Taskar et al., 2004; Riezler & Maxwell, 2006). Similarly, the best machine translation (MT) systems use discriminative methods only to calibrate the weights of a handful of different knowledge sources, which are either enumerated by hand or learned automatically but not discriminatively (e.g., Chiang, 2005). The problem with generative models is that they are typically not regularized in a principled way, and it is difficult to make up for their unregularized risk post-hoc. It is also difficult to come up with a generative model for certain kinds of data, especially the kind used to train MT systems, so approaches that rely on generative models are hard to adapt.

This paper proposes a discriminative learning method that can scale up to large structured prediction problems, without using generative models in any way. The proposed method employs the traditional AI technique of predicting a structure by searching over possible sequences of inferences, where each inference predicts a part of the eventual structure. However, unlike most approaches employed in NLP, the proposed method makes no independence assumptions: The function that evaluates each inference can use arbitrary information not only from the input, but also from all previous inferences.

Let us define some terms to help explain how our algorithm predicts a tree. An **item** is a node in the tree. Every **state** in the search space consists of a set of items, representing nodes that have been inferred since the algorithm started. States whose items form a complete tree[1] are final states. An **inference** is a (state, item) pair, i.e. a state and an item to be added to it. Each inference represents a transition from one state to another. A state is **correct** if it is possible to infer zero or more items to obtain the final state that corresponds to the training data tree. Similarly, an inference is correct if it leads to a correct state.

Given input $s$, the inference engine searches the possible complete trees $T(s)$ for the tree $\hat{t} \in T(s)$ that has minimum **cost** $C_\Theta(t)$ under model $\Theta$:

$$\hat{t} = \underset{t \in T(s)}{\arg \min} \, C_\Theta(t) = \underset{t \in T(s)}{\arg \min} \left( \sum_{j=1}^{|t|} c_\Theta(i_j) \right) \tag{1}$$

The $i_j$ are the inferences involved in constructing tree $t$. $c_\Theta(i)$ is the cost of an individual inference $i$. The number of states in the search space is exponential in the size of the input. The freedom to compute $c_\Theta(i)$ using arbitrary non-local information from anywhere in inference $i$'s state precludes exact solutions by ordinary dynamic programming. We know of two effective ways to approach such large search problems. The first, which we use for our parsing experiments, is to severely restrict the order in which items can be inferred. The second, which we use for translation, is to make the simplifying assumption that the cost of adding a given item to a state is the same for all states. Under this assumption, the fraction of any state's cost due to a particular item can be computed just once per item, instead of once per state. However, in contrast to traditional context-free parsing algorithms, that computation can involve context-sensitive features.

An important design decision in learning the inference cost function $c_\Theta$ is the choice of feature set. Given the typically very large number of possible features, the learning method must satisfy two criteria. First, it must be able to learn effectively even if the number of irrelevant features is exponential in the number of examples. It is too time-consuming to manually figure out the right feature set for such problems. Second, the learned function must be sparse. Otherwise, it would be too large for the memory of an ordinary computer, and therefore impractical.

Section 2 presents an algorithm that satisfies these criteria. This algorithm is in the family that has been shown to converge to an $\ell_1$-optimal separating hyperplane, which maximizes the minimum $\ell_1$-margin on separable training data (Rosset et al., 2004). Sections 3 and 4 present experiments on parsing and translation, respectively, illustrating the advantages of this algorithm. For lack of space, the experiments are described tersely; for details see Turian and Melamed (2006a) and Wellington et al. (2006). Also, Turian and Melamed (2006b) show how to reduce training time.

## 2 Learning Method

### 2.1 The Training Set

The training data used for both parsing and translation initially comes in the form of trees.[2] These *gold-standard* trees are used to generate training examples, each of which is a candidate inference: Starting at the initial state, we randomly choose a sequence of correct inferences that lead to the (gold-standard) final state. All the candidate inferences that can possibly follow each state in this sequence become part of the training set. The vast majority of these inferences will lead to incorrect states, which makes them negative examples. An advantage of this method of generating training examples is that it does not require a working inference engine and can be run prior to any training. A disadvantage of this approach is that it does not teach the model to recover from mistakes. We conjecture this this approach is not subject to label bias because states can "dampen" the mass they receive, as recommended by Lafferty et al. (2001).

The training set $I$ consists of training examples $i$, where each $i$ is a tuple $\langle X(i), y(i), b(i) \rangle$. $X(i)$ is a feature vector describing $i$, with each element in $\{0, 1\}$. We will use $X_f(i)$ to refer to the element of

$X(i)$ that pertains to feature $f$. $y(i) = +1$ if $i$ is correct, and $y(i) = -1$ if not. Some training examples might be more important than others, so each is given a **bias** $b(i) \in \mathbb{R}^+$. By default, all $b(i) = 1$.

*A priori*, we define only a set $A$ of simple **atomic** features (described later). The learner then induces **compound** features, each of which is a conjunction of possibly negated atomic features. Each atomic feature can have one of three values (yes/no/don't care), so the size of the compound feature space is $3^{|A|}$, exponential in the number of atomic features. In our experiments, it was also exponential in the number of training examples, because $|A| \approx |I|$. For this reason, we expect that the number of irrelevant (compound) features is exponential in the number of training examples.

## 2.2 Objective Function

The training method induces a real-valued inference evaluation function $h_\Theta(i)$. In the present work, $h_\Theta$ is a linear model parameterized by a real vector $\Theta$, which has one entry for each feature $f$:

$$h_\Theta(i) = \Theta \cdot X(i) = \sum_f \Theta_f \cdot X_f(i) \tag{2}$$

The sign of $h_\Theta(i)$ predicts the $y$-value of $i$ and the magnitude gives the confidence in this prediction. The training procedure adjusts $\Theta$ to minimize the expected risk $R_\Theta$ over training set $I$. $R_\Theta$ is the **objective** function, which is the sum of **loss** function $L_\Theta$ and **regularization** term $\Omega_\Theta$. We use the log-loss and $\ell_1$ regularization, so we have

$$R_\Theta(I) = L_\Theta(I) + \Omega_\Theta = \left[ \sum_{i \in I} l_\Theta(i) \right] + \Omega_\Theta = \left[ \sum_{i \in I} [b(i) \cdot \ln(1 + \exp(-\mu_\Theta(i)))] \right] + \left[ \lambda \cdot \sum_f |\Theta_f| \right] \tag{3}$$

$\lambda$ is a parameter that controls the strength of the regularizer and $\mu_\Theta(i) = y(i) \cdot h_\Theta(i)$ is the **margin** of example $i$. The tree cost $C_\Theta$ (Equation 1) is obtained by computing the objective function with $y(i) = +1$ and $b(i) = 1$ for every inference in the tree, and treating the penalty term $\Omega_\Theta$ as constant. I.e., $c_\Theta(i) = \ln(1 + \exp(-h_\Theta(i)))$.

This choice of objective function was motivated by Ng (2004), who showed that it is possible to achieve sample complexity that is logarithmic in the number of irrelevant features by minimizing the $\ell_1$-regularized log-loss. On the other hand, Ng showed that most other discriminative learning algorithms used for structured prediction in NLP will overfit in this setting, including: the perceptron algorithm, unregularized logistic regression, logistic regression with an $\ell_2$ penalty (a Gaussian prior), SVMs using most kernels, and neural nets trained by back-propagation.

## 2.3 Boosting $\ell_1$-Regularized Decision Trees

We use an ensemble of confidence-rated decision trees (Schapire & Singer, 1999) to represent $h_\Theta$.[3] Each internal node is split on an atomic feature. The path from the root to each node $n$ in a decision tree corresponds to a *compound* feature $f$, and we write $\varphi(n) = f$. An inference $i$ percolates down to node $n$ iff $X_{\varphi(n)} = 1$. Each leaf node $n$ keeps track of the parameter value $\Theta_{\varphi(n)}$. To score an inference $i$ using a decision tree, we percolate the inference down to a leaf $n$ and return confidence $\Theta_{\varphi(n)}$. The score $h_\Theta(i)$ given to an inference $i$ by the whole ensemble is the sum of the confidences returned by all trees in the ensemble.

---
**Listing 1** Outline of training algorithm.

| | |
|---|---|
| **procedure** TRAIN($I$) | **procedure** MAKETREE($t$, $I$) |
|     ensemble $\leftarrow \emptyset$ |     **while** some leaf in $t$ can be split **do** |
|     $\ell_1$ parameter $\lambda \leftarrow \infty$ |         split the leaf to maximize gain |
|     **while** not converged **do** |     percolate every $i \in I$ to a leaf node |
|         $t \leftarrow$ tree with one (root) node |     **for each** leaf $n$ in $t$ **do** |
|         **while** the root node cannot be split **do** |         update $\Theta_{\varphi(n)}$ to minimize $R_\Theta$ |
|             decay $\lambda$ |     append $t$ to ensemble |
|         MAKETREE($t$, $I$) | |

---

Listing 1 presents our training algorithm. At the beginning of training, the ensemble is empty, $\Theta = \mathbf{0}$, and $\lambda$ is set to $\infty$. We grow the ensemble until the objective cannot be further reduced for the current

choice of $\lambda$. We then relax the regularization penalty by decreasing $\lambda$ and continue training. In this way, instead of choosing the best $\lambda$ heuristically, we can optimize it during a single training run.

Each invocation of MakeTree has several steps. First, we choose some compound features that will allow us to decrease the objective function. We do this by building a decision tree, whose leaf node paths represent the chosen compound features. Second, we confidence-rate each leaf to minimize the objective over the examples that percolate down to that leaf. Finally, we append the decision tree to the ensemble and update parameter vector $\Theta$ accordingly. In this manner, compound feature selection is performed incrementally during training, as opposed to *a priori*.

Our strategy for feature selection is a variant of steepest descent (Perkins et al., 2003), extended to work over the *compound* feature space. The construction of each decision tree begins with a root node, which corresponds to a dummy "always true" feature. To avoid the discontinuity at $\Theta_f = 0$ of the gradient of the regularization term in the objective (Equation 3), we define the **gain** of feature $f$ as:

$$ G_\Theta(I; f) = \max\left(0, \left|\frac{\partial L_\Theta(I)}{\partial \Theta_f}\right| - \lambda\right) \tag{4} $$

The gain function indicates how the polyhedral structure of the $\ell_1$ norm tends to keep the model sparse (Riezler & Vasserman, 2004). Unless the magnitude of the gradient of the loss $|\partial L_\Theta(I)/\partial \Theta_f|$ exceeds the penalty term $\lambda$, the gain is zero and the objective cannot be reduced by adjusting parameter $\Theta_f$ away from zero. However, if the gain is non-zero, $G_\Theta(I; f)$ is the magnitude of the gradient of the objective as we adjust $\Theta_f$ in the direction that reduces $R_\Theta$. Let us define the **weight** of an example $i$ under the current model as the rate at which loss decreases as the margin of $i$ increases:

$$ w_\Theta(i) = -\frac{\partial l_\Theta(i)}{\partial \mu_\Theta(i)} = b(i) \cdot \frac{1}{1 + \exp(\mu_\Theta(i))} \tag{5} $$

Now, to compute the gain (Equation 4), we note that:

$$ \frac{\partial L_\Theta(I)}{\partial \Theta_f} = \sum_{i \in I} \frac{\partial l_\Theta(i)}{\partial \Theta_f} = \sum_{i \in I} \frac{\partial l_\Theta(i)}{\partial \mu_\Theta(i)} \cdot \frac{\partial \mu_\Theta(i)}{\partial \Theta_f} = -\sum_{i \in I} w_\Theta(i) \cdot [y(i) \cdot X_f(i)] = -\sum_{\substack{i \in I: \\ X_f(i)=1}} w_\Theta(i) \cdot y(i) \tag{6} $$

We recursively split leaf nodes by choosing the best atomic splitting feature that will allow us to increase the gain. Specifically, we consider splitting each leaf node $n$ using atomic feature $\hat{a}$, where

$$ \hat{a} = \arg\max_{a \in A} \left[G_\Theta(I; f \wedge a) + G_\Theta(I; f \wedge \neg a)\right] \tag{7} $$

Splitting using $\hat{a}$ would create children nodes $n_1$ and $n_2$, with $\varphi(n_1) = f \wedge \hat{a}$ and $\varphi(n_2) = f \wedge \neg\hat{a}$. We split node $n$ using $\hat{a}$ only if the total gain of these two children exceeds the gain of the unsplit node, i.e. if:

$$ G_\Theta(I; f \wedge \hat{a}) + G_\Theta(I; f \wedge \neg\hat{a}) > G_\Theta(I; f) \tag{8} $$

Otherwise, $n$ remains a leaf node of the decision tree, and $\Theta_{\varphi(n)}$ becomes one of the values to be optimized during the parameter update step.

Parameter update is done sequentially on only the most recently added compound features, which correspond to the leaves of the new decision tree. After the entire tree is built, we percolate each example down to its appropriate leaf node. A convenient property of decision trees is that the leaves' compound features are mutually exclusive, so their parameters can be directly optimized independently of each other. We use a line search to choose for each leaf node $n$ the parameter $\Theta_{\varphi(n)}$ that minimizes the objective over the examples in $n$.

## 3 Parsing

The parsing algorithm starts from an initial state that contains one terminal item per input word, labeled with a part-of-speech (POS) tag by the method of Ratnaparkhi (1996). For simplicity and efficiency, we impose a (deterministic) bottom-up right-to-left order for adding items to a state. The resulting search space is still exponential, and one might worry about search errors. However, in our experiments, the inference evaluation function was learned accurately enough to guide the parser to the optimal parse reasonably quickly without pruning, and thus without search errors.

Following Taskar et al. (2004), we trained and tested a parser using the algorithm in Section 2 on $\leq 15$ word sentences from the English Penn Treebank (Taylor et al., 2003). We used sections 02–21

**Table 1** Accuracy on the English Penn Treebank, training and testing on sentences of ≤ 15 words.

|  | % Recall | % Precision | $F_1$ |
|---|---|---|---|
| Turian and Melamed (2005) | 86.47 | 87.80 | 87.13 |
| Bikel (2004) | 87.85 | 88.75 | 88.30 |
| Taskar et al. (2004) | 89.10 | 89.14 | 89.12 |
| our parser | **89.26** | **89.55** | **89.40** |

for training, section 22 for development, and section 23 for testing. There were 40 million training inferences. Turian and Melamed (2005) observed that uniform example biases $b(i)$ produced lower accuracy as training progressed, because the model minimized the error per *example*. To minimize the error per *state*, we assigned every training state equal value and shared half the value uniformly among negative examples generated from that state and gave the other half to the positive examples.

Our atomic feature set $A$ contained features of the form "is there an item in group $J$ whose label/headword/headtag/headtagclass is X?". Possible values of X for each predicate were collected from the training data. Some examples of possible values for $J$ are the last $n$ child items, the first $n$ left-context items, all right-context items, and the terminal items dominated by the non-head child items. These feature templates gave rise to 1.1 million different atomic features. Significantly smaller feature sets lowered accuracy on the development set.

To situate our results in the literature, we compared them to those reported by Taskar et al. (2004) and Turian and Melamed (2005) for their discriminative parsers, which were also trained and tested on ≤ 15 word sentences.[4] We also compared our parser to a representative non-discriminative parser (Bikel, 2004)[5], the only one that we were able to train and test under exactly the same experimental conditions, including the use of POS tags from Ratnaparkhi (1996). The comparison was in terms of the standard PARSEVAL measures (Black et al., 1991): labeled precision, labeled recall, and labeled F-measure, which are based on the number of non-terminal items in the parser's output that match those in the gold-standard parse. Table 1 shows the results of these four parsers on the test set. The accuracy of our parser is at least as high as that of comparable parsers in the literature.

An advantage of our choice of loss function is that each of the binary classifiers can be learned independently of the others. We parallelized training by inducing 26 separate classifiers, one for each non-terminal label in the Penn Treebank. It took less than five CPU-days to build each of the ensembles used at test time by the final parser. By comparison, it took several CPU-months to train the parser of Taskar et al. (2004) (Dan Klein, p.c.).

## 4 Translation

The experiments in this section employed the tree transduction approach to translation, which is used by today's best MT systems (Marcu et al., 2006). To translate by tree transduction, we assume that the input sentence has already been parsed by a parser like the one described in Section 3. The transduction algorithm performs a sequence of inferences to transform this input parse tree into an output parse tree, which has words of the target language in its leaves, often in a different order than the corresponding words in the source tree. The words are then read off the target tree and outputted; the rest of the tree is discarded. Inferences are ordered by their cost, just like in ordinary parsing, and tree transduction stops when each source node has been transduced.

The data for our experiments came from the English and French components of the EuroParl corpus (Koehn, 2005). From this corpus, we extracted sentence pairs where both sentences had between 5 and 40 words, and where the ratio of their lengths was no more than 2:1. We then extracted disjoint training, tuning, development, and test sets. The tuning, development, and test sets were 1000 sentence pairs each. Typical MT systems in the literature are trained on hundreds of thousands of sentence pairs, so our main experiment used 100K sentence pairs of training data. Where noted, preliminary experiments were performed using 10K sentence pairs of training data. We computed parse trees for all the English sentences in all data sets. For each of our two training sets, we induced word alignments using the default configuration of GIZA++ (Och & Ney, 2003). The training set

word alignments and English parse trees were fed into the default French-English hierarchical alignment algorithm distributed with the GenPar system (Burbank et al., 2005) to produce binarized tree alignments. Tree alignments are the ideal form of training data for tree transducers, because they fully specify the relation between nodes in the source tree and nodes in the target tree.

We experimented with a simplistic tree transducer that involves only two types of inferences. The first type transduces words at the leaves of the source tree; the second type transduces internal nodes. To transduce a word $w$ at the leaf, the transducer replaces it with a single word $v$ that is a translation of $w$. $v$ can be empty ("NULL"). Leaves that are transduced to NULL are deterministically erased. Internal nodes are transduced merely by permuting the order of their children, where one of the possible permutations is to retain the original order. E.g., for a node with two children, the permutation classifier predicts either (1,2) or (2,1). This transducer is grossly inadequate for modeling real translations (Galley et al., 2004): It cannot account for many kinds of noise nor for many real translingual phenomena, such as head-switching and discontinuous constituents, which are important for accurate MT. It cannot even capture common "phrasal" translations such as English *there is* to French *il y a*. However, it is sufficient for controlled comparison of learning methods. One could apply the same learning methods to more sophisticated tree transducers.

When inducing leaf transducers using 10K training sentence pairs, there were 819K training inferences and 80.9K tuning inferences. For 100K training sentence pairs, there were 36.8M and 375K, respectively. And for inducing internal node transducers using 100K training sentence pairs, there were 1.0M and 9.2K, respectively. 362K leaf transduction inferences were used for development. We parallelized training of the word transducers according to the source and target word pair $(w, v)$. Prior to training, we filtered out word translation examples that were likely to be noise.[6] Given this filtering, we induced 11.6K different word transducers over 10K training sentence pairs, and 41.3K over 100K sentence pairs.

We used several kinds of features to evaluate leaf transductions. "Window" features included the source words and part-of-speech (POS) tags within a 2-word window around the word in the leaf (the "focus" word), along with their relative positions (from -2 to +2). "Co-occurrence" features included all words and POS tags from the whole source sentence, without position information. "Dependency" features were compiled from the automatically generated English parse trees. The literature on monolingual parsing gives a standard procedure for annotating each node in an English parse tree with its "lexical head word." The dependency features of each word were the label of its maximal projection[7], the label and lexical head of the parent of the maximal projection, the label and lexical head of all dependents of the maximal projection, and all the labels of all head-children, recursively, of the maximal projection. The features used to evaluate transductions of internal nodes included all those listed for leaf transduction above, where the focus words were the head words of the children of the internal node. Using these features, we applied the method of Section 2 to induce confidence-rating binary classifiers for each word pair in the lexicon, and additional binary classifiers for predicting the permutations of the children of internal tree nodes.

Before attempting the whole transduction task, we compared the model of Section 2 with the model of Vickrey et al. (2005), which learned word transduction classifiers using logistic regression with $\ell_2$ regularization. The $\ell_2$ parameters were optimized using the conjugate gradient implementation of Daumé (2004). We induced word transduction classifiers over the 10K training data using this model and our own, and tested them on the development set. The accuracy of the two models was statistically indistinguishable (about 54%). However, the models were vastly different in their size. The boosted decision trees had a total of about 38.7K non-zero compound features over an even smaller number of atomic features. In contrast, the $\ell_2$-regularized model had about 6.5 million non-zero features—an increase of more than two orders of magnitude.

We estimated that, to scale up to training data sizes typically used by modern statistical MT systems, the $\ell_2$ classifiers would not fit in memory. To make them fit, we set all but the heaviest feature weights to zero. The number of features allowed to remain active in the $\ell_2$ classifier was the number of active features in the $\ell_1$ classifier. With the playing field leveled, the accuracy of the $\ell_2$ classifiers was only

**Table 2** Accuracy of tree transducers using 100K sentence pairs of training data.

|  | exponent = 1.0 | | | exponent = 2.0 | | |
|---|---|---|---|---|---|---|
|  | Precision | Recall | $F_1$ | Precision | Recall | $F_1$ |
| generative | 51.29 | 38.30 | 43.85 | 22.62 | 16.90 | 19.35 |
| discriminative | 62.36 | 39.06 | 48.04 | 28.02 | 17.55 | 21.59 |

45%, even worse than the baseline accuracy of 48% obtained by always predicting the most common translation.

In the main experiment, we compared two models of the inference cost function $c_\Theta$—one generative and one discriminative. The generative model was a top-down tree transducer (Comon et al., 1997), which stochastically generates the target tree top-down given the source tree. Under this model, the loss of an inference $i$ is the negative log-probability of the node $n(i)$ that it infers. We estimated the parameters of this transducer using the Viterbi approximation to the inside-outside algorithm described by Graehl and Knight (2004). We lexicalized the nodes so that their probabilities could capture bilexical dependencies. Our hypothesis was that the discriminative approach would be more accurate than the generative model, because its evaluation of each inference could take into account a greater variety of information in the tree, including its entire yield (string), not just the information in nearby nodes.

We used the second search technique described in Section 1 to find the minimum cost target tree. For efficiency, we used a chart to keep track of item costs, and pruned items whose cost was more than $10^3$ times the cost of the least expensive item in the same chart cell. We also pruned items whenever the number of items in the same cell exceeded 40. Our entire tree transduction algorithm was equivalent to bottom-up synchronous parsing (Melamed, 2004) where the source side of the output bi-tree is constrained by the input (source) tree.

We compared the generative and discriminative models by reading out the string encoded in their predicted trees, and computing the F-measure between that string and the reference target sentence in the test corpus. Turian et al. (2003) show how to compute precision, recall, and the F-measure over pairs of strings without double-counting. Their family of measures is parameterized by an exponent. With the exponent set to 1.0, the F-measure is essentially the unigram overlap ratio. With the exponent set to 2.0, the F-measure rewards longer $n$-gram matches without double-counting. The generative transducer achieved its highest F-measure when the input parse trees were computed by the generative parser of Bikel (2004). The discriminatively trained transducer was most accurate when the source trees were computed by the parser in Section 3. Table 2 shows the results—the discriminatively trained transducer was much more accurate on all measures, at a statistical significance level of 0.001 using the Wilcoxon signed ranks test.

## Conclusion

We have demonstrated how to predict tree structures using binary classifiers. These classifiers are discriminatively induced by boosting confidence-rated decision trees to minimize the $\ell_1$-regularized log-loss. For large problems in tree-structured prediction, such as natural language parsing and translation, this learning algorithm has several attractive properties. It learned a purely discriminative machine over 40 million training examples and 1.1 million atomic features, using no generative model of any kind. The method did not require a great deal of feature engineering *a priori*, because it performed feature selection over a compound feature space as it learned. To our knowledge, this is the first purely discriminatively trained constituent parser that surpasses a generative baseline, as well as the first published method for purely discriminative training of a syntax-driven MT system that makes no use of generative translation models, either in training or translation. In future work, we plan to integrate the parsing and translation methods described in our experiments, to reduce compounded error.

## Acknowledgments

The authors would like to thank Léon Bottou, Patrick Haffner, Fernando Pereira, Cynthia Rudin, and the anonymous reviewers for their helpful comments and constructive criticism. This research was sponsored by NSF grants #0238406 and #0415933.

## Footnotes

[1]  What counts as a complete tree is problem-specific. E.g., in parsing, a complete tree is one that covers the input and has a root labeled TOP.

[2]  Section 4 shows how to do MT by predicting a certain kind of tree.

[3]  Turian and Melamed (2005) built more accurate parsers more quickly using decision trees rather than decision stumps, so we build full decision trees.

[4]  The results reported by Taskar et al. (2004) were not for a purely discriminative parser. Their parser beat the generative model of Bikel (2004) only after using the output from a generative model as a feature.

[5]  Bikel (2004) is a "clean room" reimplementation of the Collins (1999) model with comparable accuracy.

[6]  Specifically: $v$ was retained as a possible translation of $w$ if $v$ was the most frequent translation of $w$, or if $v$ occurred as a translation of $w$ at least three times and accounted for at least 20% of the translations of $w$ in the training data.

[7]  I.e., the highest node that has the focus word as its lexical head; if it is a leaf, then that label is a POS tag.

# References

Bikel, D. M. (2004). Intricacies of Collins' parsing model. *Computational Linguistics*, *30*(4), 479–511.

Black, E., Abney, S., Flickenger, D., Gdaniec, C., Grishman, R., Harrison, P., et al. (1991). A procedure for quantitatively comparing the syntactic coverage of English grammars. In *Speech and Natural Language.*

Burbank, A., Carpuat, M., Clark, S., Dreyer, M., Fox, P., Groves, D., et al. (2005). *Final report on statistical machine translation by parsing* (Tech. Rep.). Johns Hopkins University Center for Speech and Language Processing. `http://www.clsp.jhu.edu/ws2005/groups/statistical/report.html`.

Charniak, E., & Johnson, M. (2005). Coarse-to-fine n-best parsing and MaxEnt discriminative reranking. In *ACL.*

Chiang, D. (2005). A hierarchical phrase-based model for statistical machine translation. In *ACL.*

Collins, M. (1999). *Head-driven statistical models for natural language parsing.* Doctoral dissertation, University of Pennsylvania.

Collins, M., & Roark, B. (2004). Incremental parsing with the perceptron algorithm. In *ACL.*

Comon, H., Dauchet, M., Gilleron, R., Jacquemard, F., Lugiez, D., Tison, S., et al. (1997). *Tree automata techniques and applications.* Available at `http://www.grappa.univ-lille3.fr/tata`. (released October 1, 2002)

Cowan, B., Kučerová, I., & Collins, M. (2006). A discriminative model for tree-to-tree translation. In *EMNLP.*

Daumé, H. (2004). *Notes on CG and LM-BFGS optimization of logistic regression.* (Paper available at `http://pub.hal3.name#daume04cg-bfgs`, implementation available at `http://hal3.name/megam/`)

Galley, M., Hopkins, M., Knight, K., & Marcu, D. (2004). What's in a translation rule? In *HLT-NAACL.*

Graehl, J., & Knight, K. (2004). Training tree transducers. In *HLT-NAACL.*

Henderson, J. (2004). Discriminative training of a neural network statistical parser. In *ACL.*

Koehn, P. (2005). Europarl: A parallel corpus for statistical machine translation. In *MT Summit X.*

Lafferty, J., McCallum, A., & Pereira, F. (2001). Conditional random fields: Probabilistic models for segmenting and labeling sequence data. In *ICML.*

Marcu, D., Wang, W., Echihabi, A., & Knight, K. (2006). SPMT: Statistical machine translation with syntactified target language phrases. In *EMNLP.*

Melamed, I. D. (2004). Statistical machine translation by parsing. In *ACL.*

Ng, A. Y. (2004). Feature selection, $\ell_1$ vs. $\ell_2$ regularization, and rotational invariance. In *ICML.*

Och, F. J., & Ney, H. (2003). A systematic comparison of various statistical alignment models. *Computational Linguistics*, *29*(1), 19–51.

Perkins, S., Lacker, K., & Theiler, J. (2003). Grafting: Fast, incremental feature selection by gradient descent in function space. *Journal of Machine Learning Research*, *3*, 1333–1356.

Ratnaparkhi, A. (1996). A maximum entropy part-of-speech tagger. In *EMNLP.*

Riezler, S., & Maxwell, J. T. (2006). Grammatical machine translation. In *HLT-NAACL.*

Riezler, S., & Vasserman, A. (2004). Incremental feature selection and $\ell_1$ regularization for relaxed maximum-entropy modeling. In *EMNLP.*

Rosset, S., Zhu, J., & Hastie, T. (2004). Boosting as a regularized path to a maximum margin classifier. *Journal of Machine Learning Research*, *5*, 941–973.

Schapire, R. E., & Singer, Y. (1999). Improved boosting using confidence-rated predictions. *Machine Learning*, *37*(3), 297–336.

Taskar, B., Klein, D., Collins, M., Koller, D., & Manning, C. (2004). Max-margin parsing. In *EMNLP.*

Taylor, A., Marcus, M., & Santorini, B. (2003). The Penn Treebank: An overview. In A. Abeillé (Ed.), *Treebanks: Building and using parsed corpora* (chap. 1).

Turian, J., & Melamed, I. D. (2005). Constituent parsing by classification. In *IWPT.*

Turian, J., & Melamed, I. D. (2006a). Advances in discriminative parsing. In *ACL.*

Turian, J., & Melamed, I. D. (2006b). Computational challenges in parsing by classification. In *HLT-NAACL workshop on Computationally Hard Problems and Joint Inference in Speech and Language Processing.*

Turian, J., Shen, L., & Melamed, I. D. (2003). Evaluation of machine translation and its evaluation. In *MT Summit IX.*

Vickrey, D., Biewald, L., Teyssier, M., & Koller, D. (2005). Word-sense disambiguation for machine translation. In *EMNLP.*

Wellington, B., Turian, J., Pike, C., & Melamed, I. D. (2006). Scalable purely-discriminative training for word and tree transducers. In *AMTA.*
